# An Analog VLSI Model of Adaptation in the Vestibulo-Ocular Reflex

**Stephen P. DeWeerth and Carver A. Mead**
California Institute of Technology
Pasadena, CA 91125

## ABSTRACT

The vestibulo-ocular reflex (VOR) is the primary mechanism that controls the compensatory eye movements that stabilize retinal images during rapid head motion. The primary pathways of this system are feed-forward, with inputs from the semicircular canals and outputs to the oculomotor system. Since visual feedback is not used directly in the VOR computation, the system must exploit motor learning to perform correctly. Lisberger(1988) has proposed a model for adapting the VOR gain using image-slip information from the retina. We have designed and tested analog very large-scale integrated (VLSI) circuitry that implements a simplified version of Lisberger's adaptive VOR model.

## 1   INTRODUCTION

A characteristic commonly found in biological systems is their ability to adapt their function based on their inputs. The combination of the need for precision and the variability inherent in the environment necessitates such learning in organisms. Sensorimotor systems present obvious examples of behaviors that require learning to function correctly. Simple actions such as walking, jumping, or throwing a ball are not performed correctly the first time they are attempted; rather, they require motor learning throughout many iterations of the action.

When creating artificial systems that must execute tasks accurately in uncontrolled environments, designers can exploit adaptive techniques to improve system performance. With this in mind, it is possible for the system designer to take inspiration from systems already present in biology. In particular, sensorimotor systems, due to

their direct interfaces with the environment, can gather an immediate indication of the correctness of an action, and hence can learn without supervision. The salient characteristics of the environment are extracted by the adapting system and do not need to be specified in a user-defined training set.

## 2   THE VESTIBULO-OCULAR REFLEX

The vestibulo-ocular reflex (VOR) is an example of a sensorimotor system that requires adaptation to function correctly. The desired response of this system is a gain of −1.0 from head movements to eye movements (relative to the head), so that, as the head moves, the eyes remain fixed relative to the surroundings. Due to the feed-forward nature of the primary VOR pathways, some form of adaptation must be present to calibrate the gain of the response in infants and to maintain this calibration during growth, disease, and aging (Robinson, 1976).

Lisberger (1988) demonstrated variable gain of the VOR by fitting magnifying spectacles onto a monkey. The monkey moved about freely, allowing the VOR to learn the new relationship between head and eye movements. The monkey was then placed on a turntable, and its eye velocity was measured while head motion was generated. The eye-velocity response to head motion for three different lens magnifications is shown in Figure 1.

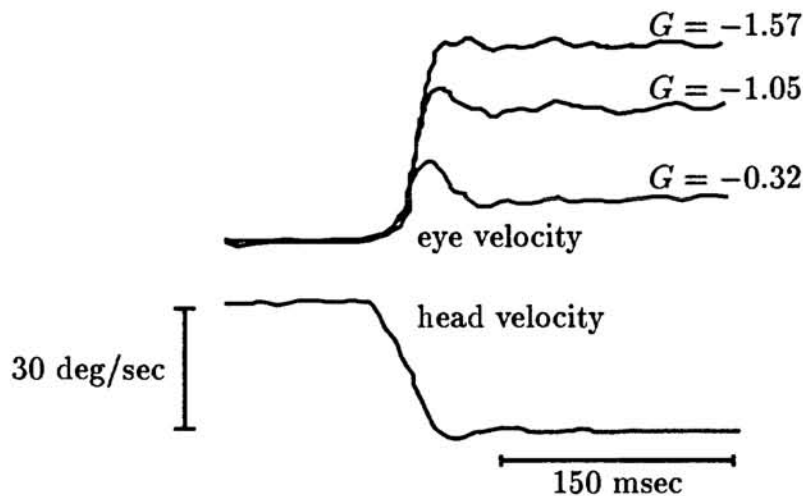

**Figure 1:** VOR data from Lisberger (1988). A monkey was fitted with magnifying spectacles and allowed to learn the gain needed for an accurate VOR. The monkey's head was then moved at a controlled velocity, and the eye velocity was measured. Three experiments were performed with spectacle magnifications of 0.25, 1.0, and 2.0. The corresponding eye velocities showed VOR gains $G$ of −0.32, −1.05, and −1.57.

Lisberger has proposed a simple model for this adaptation that uses retinal-slip information from the visual system, along with the head-motion information from the vestibular system, to adapt the gain of the forward pathways in the VOR.

Figure 2 is a schematic diagram of the pathways subserving the VOR. There are two parallel VOR pathways from the vestibular system to the motor neurons that control eye movements (Snyder, 1988). One pathway consists of vestibular inputs, VOR interneurons, and motor neurons. This pathway has been shown to exhibit an unmodified gain of approximately −0.3. The second pathway consists of vestibular inputs, floccular target neurons (FTN), and motor neurons. This pathway is the site of the proposed gain adaptation.

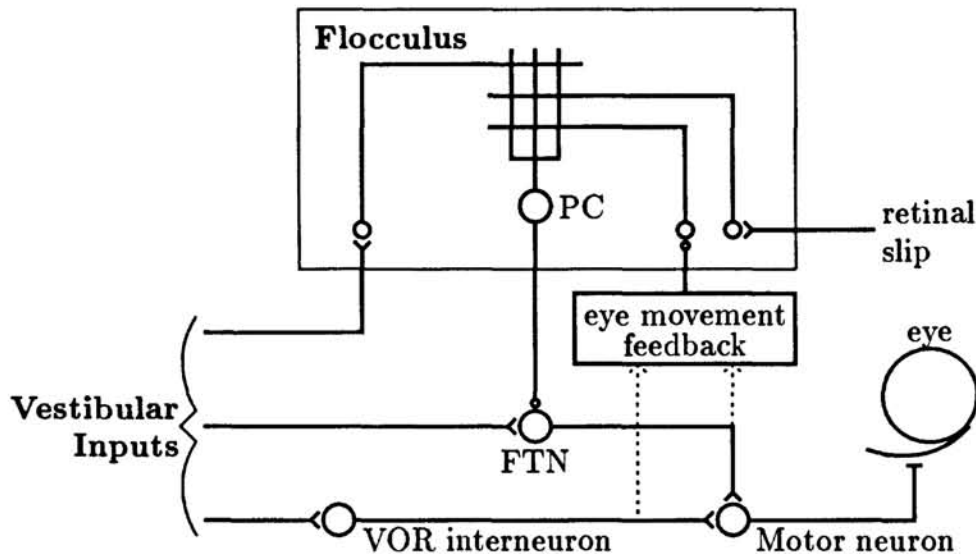

**Figure 2:** A schematic diagram of the VOR (Lisberger, 1988). Two pathways exist connecting the vestibular neurons to the motor neurons driving the eye muscles. The unmodified pathway connects via the VOR interneurons. The modified pathway (the proposed site of gain adaptation) connects via the floccular target neurons (FTN). Outputs from the Purkinje cells (PC) in the flocculus mediate gain adaptation at the FTNs.

Lisberger's hypothesis is that feedback from the visual system through the flocculus is used to facilitate the adaptation of the gain of the FTNs. Image slip on the retina indicates that the total VOR gain is not adjusted correctly. The relationship between the head motion and the image slip on the retina determines the direction in which the gain must be changed. For example, if the head is turning to the right and the retinal image slip is to the right, the eyes are turning too slowly and the gain should be increased. The direction of the gain change can be considered to be the sign of the product of head motion and retinal image slip.

## 3   THE ANALOG VLSI IMPLEMENTATION

We implemented a simplified version of Lisberger's VOR model using primarily subthreshold analog very large-scale integrated (VLSI) circuitry (Mead, 1989). We interpreted the Lisberger data to suggest that the gain of the modified pathway

varies from zero to some fixed upper limit. This assumption gives a minimum VOR gain equal to the gain of the unmodified pathway, and a maximum VOR gain equal to the sum of the unmodified pathway gain and the maximum modified pathway gain. We designed circuitry for the unmodified pathway to give an overshoot response to a step function similar to that seen in Figure 1.

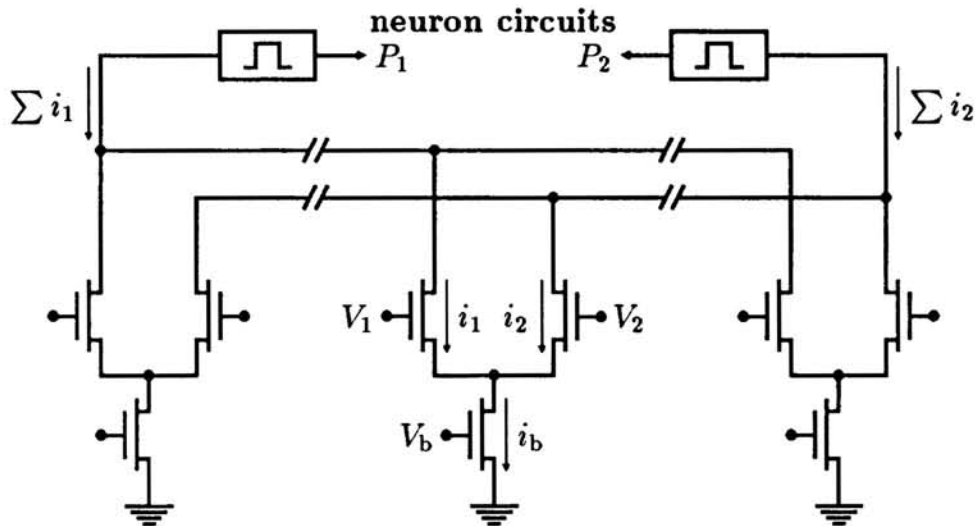

**Figure 3:** An analog VLSI sensorimotor framework. Each input circuit consists of a bias transistor and a differential pair. The voltage $V_b$ sets a fixed current $i_b$ through the bias transistor. This current is partitioned into currents $i_1$ and $i_2$ according to the differential voltage $V_1 - V_2$, and these currents are summed onto a pair of global wires. The global currents are used as inputs to two neuron circuits that convert the currents into pulse trains $P_1$ and $P_2$.

The VOR model was designed within the sensorimotor framework shown in Figure 3 (DeWeerth, 1987). The framework consists of a number of input circuits and two output circuits. Each input circuit consists of a bias transistor and a differential pair. The gain of the circuit is set by a fixed current through the bias transistor. This current is partitioned according to the differential input voltage into two currents that pass through the differential-pair transistors. The equations for these currents are

$$i_1 = i_b \frac{1}{1 + e^{V_2 - V_1}} \qquad i_2 = i_b \frac{1}{1 + e^{V_1 - V_2}}$$

The two currents are summed onto a pair of global wires. Each of these global currents is input to a neuron circuit (Mead, 1989) that converts the current linearly into the duty cycle of a pulse train. The pulse trains can be used to drive a pair of antagonistic actuators that can bidirectionally control the motion of a physical plant. We implement a system (such as the VOR) within this framework by augmenting the differential pairs with circuitry that computes the function needed for the particular application.

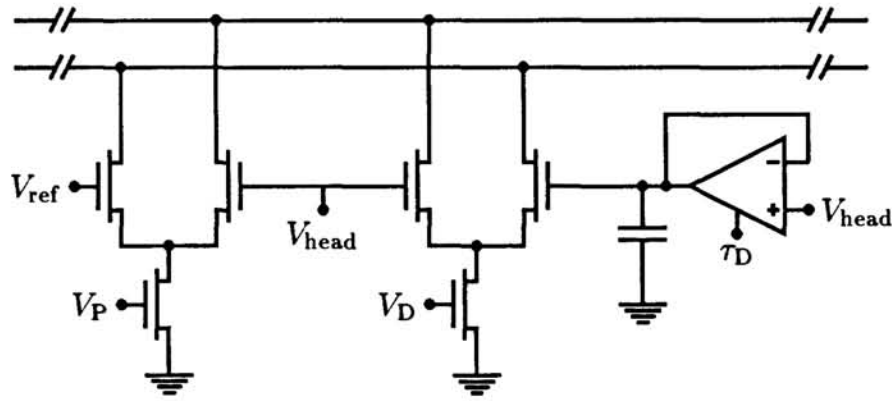

**Figure 4:** The VLSI implementation of the unmodified pathway. The left differential pair is used to convert proportionally the differential voltage representing head velocity ($V_{\text{head}} - V_{\text{ref}}$) into output currents. The right differential pair is used in conjunction with a first-order section to give output currents related to the derivative of the head velocity. The gains of the two differential pairs are set by the voltages $V_{\text{P}}$ and $V_{\text{D}}$.

The unmodified pathway is implemented in the framework using two differential pairs (Figure 4). One of these circuits proportionally converts the head motion into output currents. This circuit generates a step in eye velocity when presented with a step in head velocity. The other differential pair is combined with a first-order section to generate output currents related to the derivative of the head motion. This circuit generates a broad impulse in eye velocity when presented with a step in head velocity. By setting the gains of the proportional and derivative circuits correctly, we can make the overall response of this pathway similar to that of the unmodified pathway seen when Lisberger's monkey was presented with a step in head velocity.

We implement the modified pathway within the framework using a single differential-pair circuit that generates output currents proportional to the head velocity (Figure 5). The system adapts the gain of this pathway by integrating an error signal with respect to time. The error signal is a current, which the circuitry computes by multiplying the retinal image slip and the head velocity. This error current is integrated onto a capacitor, and the voltage on the capacitor is then converted to a current that sets the gain of the modified pathway.

## 4    EXPERIMENTAL METHOD AND RESULTS

To test our VOR circuitry, we designed a simple electrical model of the head and eye (Figure 6). The head motion is represented by a voltage that is supplied by a function generator. The oculomotor plant (the eye and corresponding muscles) is modeled by an RC circuit that integrates output pulses from the VOR circuitry into a voltage that represents eye velocity in head coordinates. We model the magnifying

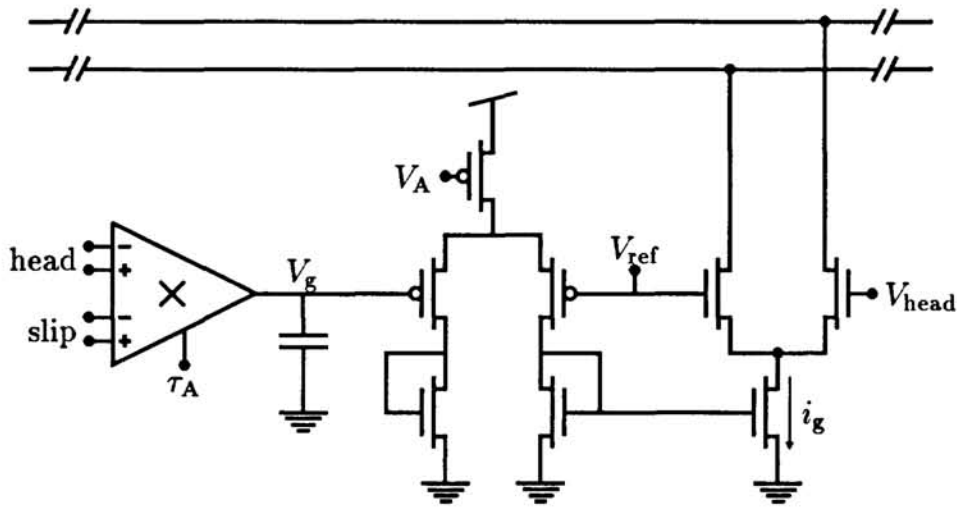

**Figure 5:** The VLSI implementation of the modified pathway. A differential pair is used to convert proportionally the differential voltage representing head velocity $(V_{\text{head}} - V_{\text{ref}})$ into output currents. Adaptive circuitry capacitively integrates the product of head velocity and retinal image slip as a voltage $V_g$. This voltage is converted to a current $i_g$ that sets the gain of the differential pair. The voltage $V_A$ sets the maximum gain of this pathway.

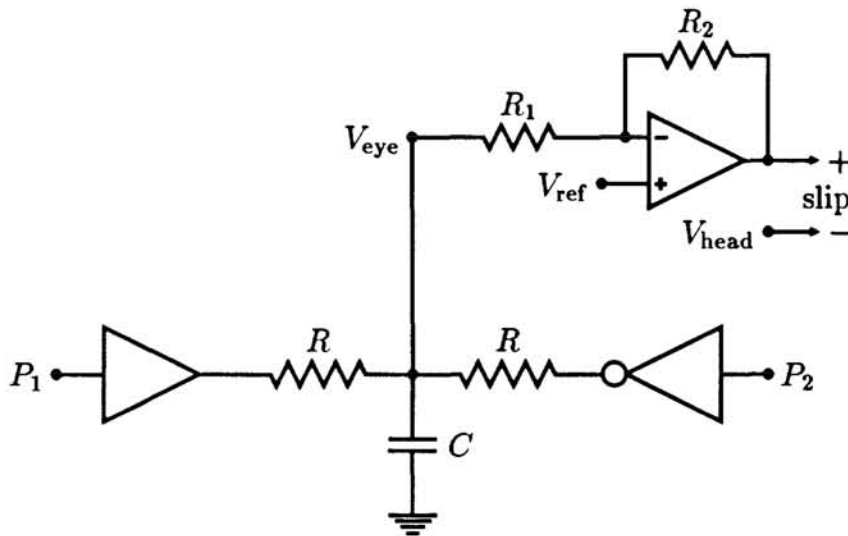

**Figure 6:** A simple model of the oculomotor plant. An RC circuit (bottom) integrates pulse trains $P_1$ and $P_2$ into a voltage $V_{\text{eye}}$ that encodes eye velocity. The magnifying spectacles are modeled by an operational amplifier circuit (top), which has a magnification $m = R_2/R_1$. The retinal image slip is encoded by the difference between the output voltage of this circuit and the voltage $V_{\text{head}}$ that encodes the head velocity.

spectacles using an operational amplifier circuit that multiplies the eye velocity by a gain before the velocity is used to compute the slip information. We compute the image slip by subtracting the head velocity from the magnified eye velocity.

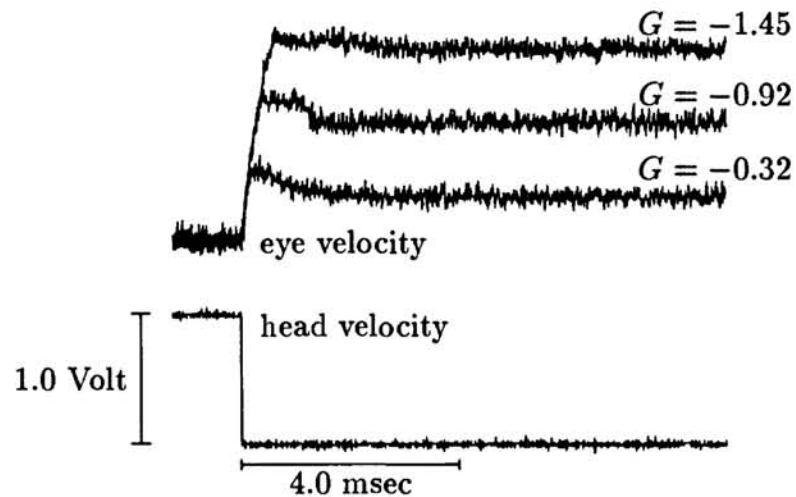

**Figure 7:** Experimental data from the VOR circuitry. The system was allowed to adapt to spectacle magnifications of 0.25, 1.0, and 2.0. After adaptation, the eye velocities showed corresponding VOR gains of $-0.32$, $-0.92$, and $-1.45$.

We performed an experiment to generate data to compare to the data measured by Lisberger (Figure 1). A head-velocity step was supplied by a function generator and was used as input to the VOR circuitry. The VOR outputs were then converted to an eye velocity by the model of the oculomotor plant. The proportional, derivative, and maximum adaptive gains were set to give a system response similar to that observed in the monkey. The system was allowed to adapt over a number of presentations of the input for each spectacle magnification. The resulting eye velocity data are displayed in Figure 7.

## 5   CONCLUSIONS AND FUTURE WORK

In this paper, we have presented an analog VLSI implementation of a model of a biological sensorimotor system. The system performs unsupervised learning using signals generated as the system interacts with its environment. This model can be compared to traditional adaptive control schemes (Åström, 1987) for performing similar tasks. In the future, we hope to extend the model presented here to incorporate more of the information known about the VOR.

We are currently designing and testing chips that use ultraviolet storage techniques for gain adaptation. These chips will allow us to achieve adaptive time constants of the same order as those found in biological systems (minutes to hours).

We are also combining our chips with a mechanical model of the head and eyes to give more accurate environmental feedback. We can acquire true image-slip data using a vision chip (Tanner, 1986) that computes global field motion.

## Acknowledgments

We thank Steven Lisberger for his suggestions for improving our implementation of the VOR model. We would also like to thank Massimo Sivilotti, Michelle Mahowald, Michael Emerling, Nanette Boden, Richard Lyon, and Tobias Delbrück for their help during the writing of this paper.

## References

K.J. Åström, Adaptive feedback control. *Proceedings of the IEEE*, 75:2:185–217, 1987.

S.P. DeWeerth, *An Analog VLSI Framework for Motor Control.* M.S. Thesis, Department of Computer Science, California Institute of Technology, Pasadena, CA, 1987.

S.G. Lisberger, The neural basis for learning simple motor skills. *Science*, 242:728–735, 1988.

C.A. Mead, **Analog VLSI and Neural Systems.** Addison-Wesley, Reading, MA, 1989.

D.A. Robinson, Adaptive gain control of vestibulo-ocular reflex by the cerebellum. *J. Neurophysiology*, 39:954–969, 1976.

L.H. Snyder and W.M. King, Vertical vestibuloocular reflex in cat: asymmetry and adaptation. *J. Neurophysiology*, 59:279–298, 1988.

J.E. Tanner. *Integrated Optical Motion Detection.* Ph.D. Thesis, Department of Computer Science, California Institute of Technology, S223:TR:86, Pasadena, CA, 1986.